# A General and *Efficient* Multiple Kernel Learning Algorithm

**Sören Sonnenburg**[*]
Fraunhofer FIRST
Kekuléstr. 7
12489 Berlin
Germany
sonne@first.fhg.de

**Gunnar Rätsch**
Friedrich Miescher Lab
Max Planck Society
Spemannstr. 39
Tübingen, Germany
raetsch@tue.mpg.de

**Christin Schäfer**
Fraunhofer FIRST
Kekuléstr. 7
12489 Berlin
Germany
christin@first.fhg.de

## Abstract

While classical kernel-based learning algorithms are based on a single kernel, in practice it is often desirable to use multiple kernels. Lankriet et al. (2004) considered conic combinations of kernel matrices for classification, leading to a convex quadratically constraint quadratic program. We show that it can be rewritten as a semi-infinite linear program that can be efficiently solved by recycling the standard SVM implementations. Moreover, we generalize the formulation and our method to a larger class of problems, including regression and one-class classification. Experimental results show that the proposed algorithm helps for automatic model selection, improving the interpretability of the learning result and works for hundred thousands of examples or hundreds of kernels to be combined.

## 1  Introduction

Kernel based methods such as Support Vector Machines (SVMs) have proven to be powerful for a wide range of different data analysis problems. They employ a so-called kernel function $\mathbf{k}(\mathbf{x}_i, \mathbf{x}_j)$ which intuitively computes the similarity between two examples $\mathbf{x}_i$ and $\mathbf{x}_j$. The result of SVM learning is a $\boldsymbol{\alpha}$-weighted linear combination of kernel elements and the bias $b$:

$$f(\mathbf{x}) = \text{sign}\left(\sum_{i=1}^{N} \alpha_i y_i \mathbf{k}(\mathbf{x}_i, \mathbf{x}) + b\right), \tag{1}$$

where the $\mathbf{x}_i$'s are $N$ labeled training examples ($y_i \in \{\pm 1\}$).

Recent developments in the literature on the SVM and other kernel methods have shown the need to consider multiple kernels. This provides flexibility, and also reflects the fact that typical learning problems often involve multiple, heterogeneous data sources. While this so-called "multiple kernel learning" (MKL) problem can in principle be solved via cross-validation, several recent papers have focused on more efficient methods for multiple kernel learning [4, 5, 1, 7, 3, 9, 2].

One of the problems with kernel methods compared to other techniques is that the resulting decision function (1) is hard to interpret and, hence, is difficult to use in order to extract rel-

---

[*]For more details, datasets and pseudocode see `http://www.fml.tuebingen.mpg.de/raetsch/projects/mkl_silp`.

evant knowledge about the problem at hand. One can approach this problem by considering convex combinations of $K$ kernels, i.e.

$$\mathbf{k}(\mathbf{x}_i, \mathbf{x}_j) = \sum_{k=1}^{K} \beta_k \mathbf{k}_k(\mathbf{x}_i, \mathbf{x}_j) \qquad (2)$$

with $\beta_k \geq 0$ and $\sum_k \beta_k = 1$, where each kernel $\mathbf{k}_k$ uses only a distinct set of features of each instance. For appropriately designed sub-kernels $\mathbf{k}_k$, the optimized combination coefficients can then be used to understand which features of the examples are of importance for discrimination: if one would be able to obtain an accurate classification by a *sparse* weighting $\beta_k$, then one can quite easily interpret the resulting decision function. We will illustrate that the considered MKL formulation provides useful insights and is at the same time is very efficient. This is an important property missing in current kernel based algorithms.

We consider the framework proposed by [7], which results in a convex optimization problem - a quadratically-constrained quadratic program (QCQP). This problem is more challenging than the standard SVM QP, but it can in principle be solved by general-purpose optimization toolboxes. Since the use of such algorithms will only be feasible for small problems with few data points and kernels, [1] suggested an algorithm based on sequential minimization optimization (SMO) [10]. While the kernel learning problem is convex, it is also non-smooth, making the direct application of simple local descent algorithms such as SMO infeasible. [1] therefore considered a smoothed version of the problem to which SMO can be applied.

In this work we follow a different direction: We reformulate the problem as a semi-infinite linear program (SILP), which can be efficiently solved using an off-the-shelf LP solver and a standard SVM implementation (cf. Section 2 for details). Using this approach we are able to solve problems with more than hundred thousand examples or with several hundred kernels quite efficiently. We have used it for the analysis of sequence analysis problems leading to a better understanding of the biological problem at hand [16, 13]. We extend our previous work and show that the transformation to a SILP works with a large class of convex loss functions (cf. Section 3). Our column-generation based algorithm for solving the SILP works by repeatedly using an algorithm that can efficiently solve the single kernel problem in order to solve the MKL problem. Hence, if there exists an algorithm that solves the simpler problem efficiently (like SVMs), then our new algorithm can efficiently solve the multiple kernel learning problem.

We conclude the paper by illustrating the usefulness of our algorithms in several examples relating to the interpretation of results and to automatic model selection.

## 2   Multiple Kernel Learning for Classification using SILP

In the Multiple Kernel Learning (MKL) problem for binary classification one is given $N$ data points $(\mathbf{x}_i, y_i)$ ($y_i \in \{\pm 1\}$), where $\mathbf{x}_i$ is translated via a mapping $\Phi_k(\mathbf{x}) \mapsto \mathbb{R}^{D_k}$, $k = 1 \dots K$ from the input into $K$ feature spaces $(\Phi_1(\mathbf{x}_i), \dots, \Phi_K(\mathbf{x}_i))$ where $D_k$ denotes the dimensionality of the $k$-th feature space. Then one solves the following optimization problem [1], which is equivalent to the linear SVM for $K = 1$:[1]

$$\min_{\mathbf{w}_k \in \mathbb{R}^{D_k}, \boldsymbol{\xi} \in \mathbb{R}^N_+, \boldsymbol{\beta} \in \mathbb{R}^K_+, b \in \mathbb{R}} \quad \frac{1}{2}\left(\sum_{k=1}^{K} \beta_k \|\mathbf{w}_k\|_2\right)^2 + C\sum_{i=1}^{N} \xi_i \qquad (3)$$

$$\text{s.t.} \quad y_i\left(\sum_{k=1}^{K} \beta_k \mathbf{w}_k^\top \Phi_k(\mathbf{x}_i) + b\right) \geq 1 - \xi_i \text{ and } \sum_{k=1}^{K} \beta_k = 1.$$

Note that the $\ell_1$-norm of $\boldsymbol{\beta}$ is constrained to one, while one is penalizing the $\ell_2$-norm of $\mathbf{w}_k$ in each block $k$ separately. The idea is that $\ell_1$-norm constrained or penalized variables tend to have sparse optimal solutions, while $\ell_2$-norm penalized variables do not [11]. Thus the above optimization problem offers the possibility to find sparse solutions on the block level with non-sparse solutions within the blocks.

Bach et al. [1] derived the dual for problem (3), which can be equivalently written as:

$$\min_{\gamma \in \mathbb{R}, \mathbf{1}C \geq \boldsymbol{\alpha} \in \mathbb{R}_+^N} \gamma \quad \text{s.t.} \quad \underbrace{\frac{1}{2} \sum_{i,j=1}^N \alpha_i \alpha_j y_i y_j \mathbf{k}_k(\mathbf{x}_i, \mathbf{x}_j) - \sum_{i=1}^N \alpha_i}_{=:S_k(\boldsymbol{\alpha})} \leq \gamma \ \text{and} \ \sum_{i=1}^N \alpha_i y_i = 0 \quad (4)$$

for $k = 1, \ldots, K$, where $\mathbf{k}_k(\mathbf{x}_i, \mathbf{x}_j) = (\Phi_k(\mathbf{x}_i), \Phi_k(\mathbf{x}_j))$. Note that we have one quadratic constraint per kernel ($S_k(\boldsymbol{\alpha}) \leq \gamma$). In the case of $K = 1$, the above problem reduces to the original SVM dual.

In order to solve (4), one may solve the following saddle point problem (Lagrangian):

$$\mathcal{L} := \gamma + \sum_{k=1}^K \beta_k (S_k(\boldsymbol{\alpha}) - \gamma) \qquad (5)$$

minimized w.r.t. $\boldsymbol{\alpha} \in \mathbb{R}_+^N, \gamma \in \mathbb{R}$ (subject to $\boldsymbol{\alpha} \leq C\mathbf{1}$ and $\sum_i \alpha_i y_i = 0$) and maximized w.r.t. $\boldsymbol{\beta} \in \mathbb{R}_+^K$. Setting the derivative w.r.t. to $\gamma$ to zero, one obtains the constraint $\sum_k \beta_k = 1$ and (5) simplifies to: $\mathcal{L} = S(\boldsymbol{\alpha}, \boldsymbol{\beta}) := \sum_{k=1}^K \beta_k S_k(\boldsymbol{\alpha})$ and leads to a min-max problem:

$$\max_{\boldsymbol{\beta} \in \mathbb{R}_+^k} \min_{\mathbf{1}C \geq \boldsymbol{\alpha} \in \mathbb{R}_+^N} \quad \sum_{k=1}^K \beta_k S_k(\boldsymbol{\alpha}) \quad \text{s.t.} \quad \sum_{i=1}^N \alpha_i y_i = 0 \ \text{and} \ \sum_{k=1}^K \beta_k = 1. \qquad (6)$$

Assume $\boldsymbol{\alpha}^*$ would be the optimal solution, then $\theta^* := S(\boldsymbol{\alpha}^*, \boldsymbol{\beta})$ is minimal and, hence, $S(\boldsymbol{\alpha}, \boldsymbol{\beta}) \geq \theta^*$ for all $\boldsymbol{\alpha}$ (subject to the above constraints). Hence, finding a saddle-point of (5) is equivalent to solving the following semi-infinite linear program:

$$\max_{\theta \in \mathbb{R}, \boldsymbol{\beta} \in \mathbb{R}_+^M} \quad \theta \quad \text{s.t.} \quad \sum_k \beta_k = 1 \ \text{and} \ \sum_{k=1}^K \beta_k S_k(\boldsymbol{\alpha}) \geq \theta \qquad (7)$$

$$\text{for all } \boldsymbol{\alpha} \text{ with } \mathbf{0} \leq \boldsymbol{\alpha} \leq C\mathbf{1} \text{ and } \sum_i y_i \alpha_i = 0$$

Note that this is a linear program, as $\theta$ and $\boldsymbol{\beta}$ are only linearly constrained. However there are infinitely many constraints: one for each $\boldsymbol{\alpha} \in \mathbb{R}^N$ satisfying $0 \leq \boldsymbol{\alpha} \leq C$ and $\sum_{i=1}^N \alpha_i y_i = 0$. Both problems (6) and (7) have the same solution. To illustrate that, consider $\boldsymbol{\beta}$ is fixed and we maximize $\boldsymbol{\alpha}$ in (6). Let $\boldsymbol{\alpha}^*$ be the solution that maximizes (6). Then we can decrease the value of $\theta$ in (7) as long as no $\boldsymbol{\alpha}$-constraint (7) is violated, i.e. down to $\theta = \sum_{k=1}^K \beta_k S_k(\boldsymbol{\alpha}^*)$. Similarly, as we increase $\theta$ for a fixed $\boldsymbol{\alpha}$ the maximizing $\boldsymbol{\beta}$ is found. We will discuss in Section 4 how to solve such semi infinite linear programs.

## 3 Multiple Kernel Learning with General Cost Functions

In this section we consider the more general class of MKL problems, where one is given an *arbitrary* strictly convex differentiable loss function, for which we derive its MKL SILP formulation. We will then investigate in this general MKL SILP using different loss functions, in particular the soft-margin loss, the $\epsilon$-insensitive loss and the quadratic loss.

We define the MKL primal formulation for a strictly convex and differentiable loss function $L$ as: (for simplicity we omit a bias term)

$$\min_{\mathbf{w}_k \in \mathbb{R}^{D_k}} \frac{1}{2} \left( \sum_{k=1}^K \|\mathbf{w}_k\| \right)^2 + \sum_{i=1}^N L(f(\mathbf{x}_i), y_i) \quad \text{s.t.} \quad f(\mathbf{x}_i) = \sum_{k=1}^K (\Phi_k(\mathbf{x}_i), \mathbf{w}_k) \qquad (8)$$

In analogy to [1] we treat problem (8) as a second order cone program (SOCP) leading to the following dual (see Supplementary Website or [17] for details):

$$\min_{\gamma \in \mathbb{R}, \boldsymbol{\alpha} \in \mathbb{R}^N} \quad \gamma - \sum_{i=1}^{N} L(L'^{-1}(\alpha_i, y_i), y_i) + \sum_{i=1}^{N} \alpha_i L'^{-1}(\alpha_i, y_i) \tag{9}$$

$$\text{s.t. :} \quad \frac{1}{2} \left\| \sum_{i=1}^{N} \alpha_i \Phi_k(\mathbf{x}_i) \right\|_2^2 \leq \gamma, \ \forall k = 1 \ldots K$$

To derive the SILP formulation we follow the same recipe as in Section 2: deriving the Lagrangian leads to a max-min problem formulation to be eventually reformulated to a SILP:

$$\max_{\theta \in \mathbb{R}, \boldsymbol{\beta} \in \mathbb{R}^K} \theta \quad \text{s.t.} \quad \sum_{k=1}^{K} \beta_k = 1 \quad \text{and} \quad \sum_{k=1}^{K} \beta_k S_k(\boldsymbol{\alpha}) \geq \theta, \ \forall \boldsymbol{\alpha} \in \mathbb{R}^N,$$

where $S_k(\boldsymbol{\alpha}) = -\sum_{i=1}^{N} L(L'^{-1}(\alpha_i, y_i), y_i) + \sum_{i=1}^{N} \alpha_i L'^{-1}(\alpha_i, y_i) + \frac{1}{2} \left\| \sum_{i=1}^{N} \alpha_i \Phi_k(\mathbf{x}_i) \right\|_2^2 .$

We assumed that $L(x, y)$ is strictly convex and differentiable in $x$. Unfortunately, the soft margin and $\epsilon$-insensitive loss do not have these properties. We therefore consider them separately in the sequel.

**Soft Margin Loss** We use the following loss in order to approximate the soft margin loss: $L_\sigma(x, y) = \frac{C}{\sigma} \log(1 + \exp((1 - xy)\sigma))$. It is easy to verify that $\lim_{\sigma \to \infty} L_\sigma(x, y) = C(1 - xy)_+$. Moreover, $L_\sigma$ is strictly convex and differentiable for $\sigma < \infty$. Using this loss and assuming $y_i \in \{\pm 1\}$, we obtain :

$$S_k(\boldsymbol{\alpha}) = -\sum_{i=1}^{N} \frac{C}{\sigma} \left( \log \left( \frac{C y_i}{\alpha_i + C y_i} \right) + \log \left( -\frac{\alpha_i}{\alpha_i + C y_i} \right) \right) + \sum_{i=1}^{N} \alpha_i y_i + \frac{1}{2} \left\| \sum_{i=1}^{N} \alpha_i \Phi_k(\mathbf{x}_i) \right\|_2^2 .$$

If $\sigma \to \infty$, then the first two terms vanish provided that $-C \leq \alpha_i \leq 0$ if $y_i = 1$ and $0 \leq \alpha_i \leq C$ if $y_i = -1$. Substituting $\alpha = -\tilde{\alpha}_i y_i$, we then obtain $S_k(\tilde{\boldsymbol{\alpha}}) = -\sum_{i=1}^{N} \tilde{\alpha}_i + \frac{1}{2} \left\| \sum_{i=1}^{N} \tilde{\alpha}_i y_i \Phi_k(\mathbf{x}_i) \right\|_2^2$, with $0 \leq \tilde{\alpha}_i \leq C$ ($i = 1, \ldots, N$), which is very similar to (4): only the $\sum_i \alpha_i y_i = 0$ constraint is missing, since we omitted the bias.

**One-Class Soft Margin Loss** The one-class SVM soft margin (e.g. [15]) is very similar to the two class case and leads to $S_k(\boldsymbol{\alpha}) = \frac{1}{2} \left\| \sum_{i=1}^{N} \alpha_i \Phi_k(\mathbf{x}_i) \right\|_2^2$ subject to $\mathbf{0} \leq \boldsymbol{\alpha} \leq \frac{1}{\nu N} \mathbf{1}$ and $\sum_{i=1}^{N} \alpha_i = 1$.

**$\epsilon$-insensitive Loss** Using the same technique for the epsilon insensitive loss $L(x, y) = C(1 - |x - y|)_+$, we obtain

$$S_k(\boldsymbol{\alpha}, \boldsymbol{\alpha}^*) = \frac{1}{2} \left\| \sum_{i=1}^{N} (\alpha_i - \alpha_i^*) \Phi_k(\mathbf{x}_i) \right\|_2^2 - \sum_{i=1}^{N} (\alpha_i + \alpha_i^*)\epsilon - \sum_{i=1}^{N} (\alpha_i - \alpha_i^*) y_i,$$

with $\mathbf{0} \leq \boldsymbol{\alpha}, \boldsymbol{\alpha}^* \leq C\mathbf{1}$. When including a bias term, we additionally have the constraint $\sum_{i=1}^{N} (\alpha_i - \alpha_i^*) y_i = 0$.

It is straightforward to derive the dual problem for other loss functions such as the quadratic loss. Note that the dual SILP's only differ in the definition of $S_k$ and the domains of the $\boldsymbol{\alpha}$'s.

## 4 Algorithms to solve SILPs

The SILPs considered in this work all have the following form:

$$\max_{\theta \in \mathbb{R}, \boldsymbol{\beta} \in \mathbb{R}^M_+} \theta \quad \text{s.t.} \quad \sum_{k=1}^{K} \beta_k = 1 \quad \text{and} \quad \sum_{k=1}^{M} \beta_k S_k(\boldsymbol{\alpha}) \geq \theta \text{ for all } \boldsymbol{\alpha} \in \mathcal{C} \tag{10}$$

for some appropriate $S_k(\boldsymbol{\alpha})$ and the feasible set $\mathcal{C} \subseteq \mathbb{R}^\mathcal{N}$ of $\boldsymbol{\alpha}$ depending on the choice of the cost function. Using Theorem 5 in [12] one can show that the above SILP has a solution if the corresponding primal is feasible and bounded. Moreover, there is no duality gap, if $\mathcal{M} = \mathrm{co}\{[S_1(\boldsymbol{\alpha}), \dots, S_K(\boldsymbol{\alpha})]^\top \mid \boldsymbol{\alpha} \in \mathcal{C}\}$ is a closed set. For all loss functions considered in this paper this holds true. We propose to use a technique called Column Generation to solve (10). The basic idea is to compute the optimal $(\boldsymbol{\beta}, \theta)$ in (10) for a restricted subset of constraints. It is called the *restricted master problem*. Then a second algorithm generates a new constraint determined by $\boldsymbol{\alpha}$. In the best case the other algorithm finds the constraint that maximizes the constraint violation for the given intermediate solution $(\boldsymbol{\beta}, \theta)$, i.e.

$$\boldsymbol{\alpha}_{\boldsymbol{\beta}} := \underset{\boldsymbol{\alpha} \in \mathcal{C}}{\mathrm{argmin}} \sum_k \beta_k S_k(\boldsymbol{\alpha}). \tag{11}$$

If $\boldsymbol{\alpha}_{\boldsymbol{\beta}}$ satisfies the constraint $\sum_{k=1}^{K} \beta_k S_k(\boldsymbol{\alpha}_{\boldsymbol{\beta}}) \geq \theta$, then the solution is optimal. Otherwise, the constraint is added to the set of constraints.

Algorithm 1 is a special case of the set of SILP algorithms known as **exchange methods**. These methods are known to converge (cf. Theorem 7.2 in [6]). However, no convergence rates for such algorithm are so far known.[2] Since it is often sufficient to obtain an approximate solution, we have to define a suitable convergence criterion. Note that the problem is solved when all constraints are satisfied. Hence, it is a natural choice to use the normalized maximal constraint violation as a convergence criterion, i.e. $\epsilon := \left| 1 - \frac{\sum_{k=1}^{K} \beta_k^t S_k(\boldsymbol{\alpha}^t)}{\theta^t} \right|$, where $(\boldsymbol{\beta}^t, \theta^t)$ is the optimal solution at iteration $t - 1$ and $\boldsymbol{\alpha}^t$ corresponds to the newly found maximally violating constraint of the next iteration.

We need an algorithm to identify unsatisfied constraints, which, fortunately, turns out to be particularly simple. Note that (11) is for all considered cases exactly the dual optimization problem of the single kernel case for fixed $\boldsymbol{\beta}$. For instance for binary classification, (11) reduces to the standard SVM dual using the kernel $\mathbf{k}(\mathbf{x}_i, \mathbf{x}_j) = \sum_k \beta_k \mathbf{k}_k(\mathbf{x}_i, \mathbf{x}_j)$:

$$\min_{\boldsymbol{\alpha} \in \mathbb{R}^N} \sum_{i,j=1}^{N} \alpha_i \alpha_j y_i y_j \mathbf{k}(\mathbf{x}_i, \mathbf{x}_j) - \sum_{i=1}^{N} \alpha_i \quad \text{with} \quad \mathbf{0} \leq \boldsymbol{\alpha} \leq C\mathbf{1} \text{ and } \sum_{i=1}^{N} \alpha_i y_i = 0.$$

We can therefore use a standard SVM implementation in order to identify the most violated constraint. Since there exist a large number of efficient algorithms to solve the single kernel problems for all sorts of cost functions, we have therefore found an easy way to extend their applicability to the problem of Multiple Kernel Learning. In some cases it is possible to extend existing SMO based implementations to simultaneously optimize $\boldsymbol{\beta}$ and $\boldsymbol{\alpha}$. In [16] we have considered such an algorithm for the binary classification case that frequently recomputes the $\boldsymbol{\beta}$'s.[3] Empirically it is a few times faster than the column generation algorithm, but it is on the other hand much harder to implement.

## 5 Experiments

In this section we will discuss toy examples for binary classification and regression, demonstrating that MKL can recover information about the problem at hand, followed by a brief review on problems for which MKL has been successfully used.

### 5.1 Classifications

In Figure 1 we consider a binary classification problem, where we used MKL-SVMs with five RBF-kernels with different widths, to distinguish the dark star-like shape from the

**Algorithm 1** The column generation algorithm employs a linear programming solver to iteratively solve the semi-infinite linear optimization problem (10). The accuracy parameter $\epsilon$ is a parameter of the algorithm. $S_k(\boldsymbol{\alpha})$ and $\mathcal{C}$ are determined by the cost function.

---

$S^0 = 1, \theta^1 = -\infty, \beta_k^1 = \frac{1}{K}$ for $k = 1, \ldots, K$

**for** $t = 1, 2, \ldots$ **do**

Compute $\boldsymbol{\alpha}^t = \underset{\boldsymbol{\alpha} \in \mathcal{C}}{\operatorname{argmin}} \sum_{k=1}^{K} \beta_k^t S_k(\boldsymbol{\alpha})$ by single kernel algorithm with $K = \sum_{k=1}^{K} \beta_k^t K_k$

$S^t = \sum_{k=1}^{K} \beta_k^t S_k(\boldsymbol{\alpha}^t)$

**if** $|1 - \dfrac{S^t}{\theta^t}| \leq \epsilon$ **then break**

$(\boldsymbol{\beta}^{t+1}, \theta^{t+1}) = \operatorname{argmax} \theta$

w.r.t. $\boldsymbol{\beta} \in \mathbb{R}_+^K, \theta \in \mathbb{R}$ with $\sum_{k=1}^{K} \beta_k = 1$ and $\sum_{k=1}^{K} \beta_k S_k^r \geq \theta$ for $r = 1, \ldots, t$

**end for**

---

light star. (The distance between the stars increases from left to right.) Shown are the obtained kernel weightings for the five kernels and the test error which quickly drops to zero as the problem becomes separable. Note that the RBF kernel with largest width was not appropriate and thus never chosen. Also with increasing distance between the stars kernels with greater widths are used. This illustrates that MKL one can indeed recover such tendencies.

## 5.2 Regression

We applied the newly derived MKL support vector regression formulation, to the task of learning a sine function using three RBF-kernels with different widths. We then increased the frequency of the sine wave. As can be seen in Figure 2, MKL-SV regression abruptly switches to the width of the RBF-kernel fitting the regression problem best. In another regression experiment, we combined a linear function with two sine waves, one of lower frequency and one of high frequency, i.e. $f(x) = c \cdot x + \sin(ax) + \sin(bx)$. Using ten RBF-kernels of different width (see Figure 3) we trained a MKL-SVR and display the learned weights (a column in the figure). The largest selected width (100) models the linear component (since RBF with large widths are effectively linear) and the medium width (1) corresponds to the lower frequency sine. We varied the frequency of the high frequency sine wave from low to high (left to right in the figure). One observes that MKL determines

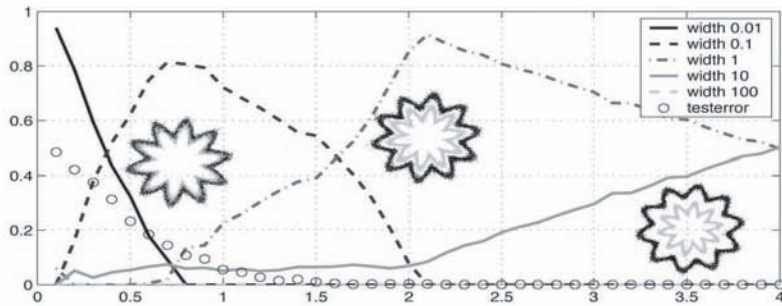

Figure 1: A 2-class toy problem where the dark grey star-like shape is to be distinguished from the light grey star inside of the dark grey star. For details see text.

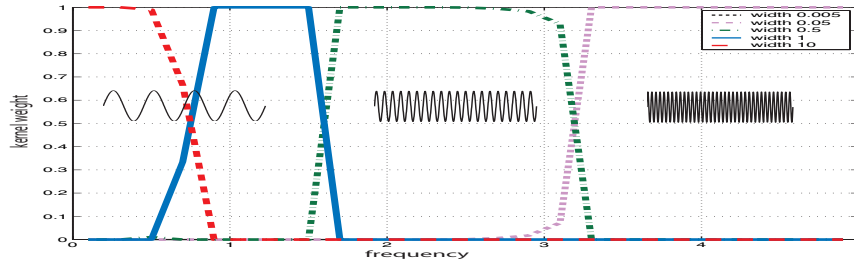

Figure 2: MKL-Support Vector Regression for the task of learning a sine wave (please see text for details).

an appropriate combination of kernels of low and high widths, while decreasing the RBF-kernel width with increased frequency. This shows that MKL can be more powerful than cross-validation: To achieve a similar result with cross-validation one has to use 3 nested loops to tune 3 RBF-kernel sigmas, e.g. train $10 \cdot 9 \cdot 8/6 = 120$ SVMs, which in preliminary experiments was much slower then using MKL (800 vs. 56 seconds).

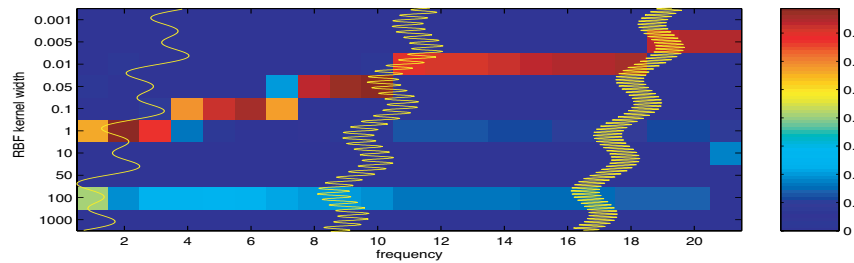

Figure 3: MKL support vector regression on a linear combination of three functions: $f(x) = c \cdot x + \sin(ax) + \sin(bx)$. MKL recovers that the original function is a combination of functions of low and high complexity. For more details see text.

## 5.3 Applications in the Real World

MKL has been successfully used on real-world datasets in the field of computational biology [7, 16]. It was shown to improve classification performance on the task of ribosomal and membrane protein prediction, where a weighting over different kernels each corresponding to a different feature set was learned. Random channels obtained low kernel weights. Moreover, on a splice site recognition task we used MKL as a tool for interpreting the SVM classifier [16], as is displayed in Figure 4. Using specifically optimized string kernels, we were able to solve the classification MKL SILP for $N = 1.000.000$ examples and $K = 20$ kernels, as well as for $N = 10.000$ examples and $K = 550$ kernels.

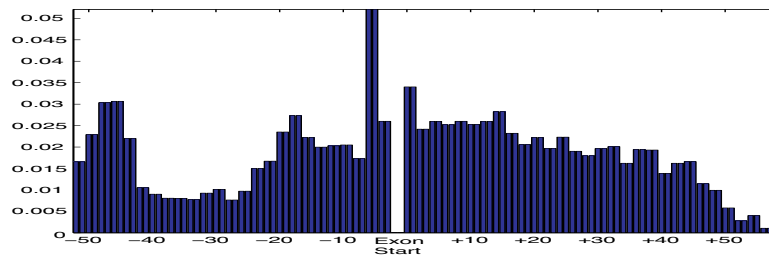

Figure 4: The figure shows an importance weighting for each position in a DNA sequence (around a so called splice site). MKL was used to learn these weights, each corresponding to a sub-kernel which uses information at that position to discriminate true splice sites from fake ones. Different peaks correspond to different biologically known signals (see [16] for details). We used 65.000 examples for training with 54 sub-kernels.

# 6 Conclusion

We have proposed a simple, yet efficient algorithm to solve the multiple kernel learning problem for a large class of loss functions. The proposed method is able to exploit the existing single kernel algorithms, whereby extending their applicability. In experiments we have illustrated that the MKL for classification and regression can be useful for automatic model selection and for obtaining comprehensible information about the learning problem at hand. It is future work to evaluate MKL algorithms for unsupervised learning such as Kernel PCA and one-class classification.

### Acknowledgments

The authors gratefully acknowledge partial support from the PASCAL Network of Excellence (EU #506778), DFG grants JA 379 / 13-2 and MU 987/2-1. We thank Guido Dornhege, Olivier Chapelle, Olaf Weiss, Joaquin Quiñoñero Candela, Sebastian Mika and K.-R. Müller for great discussions.

## Footnotes

[1][1] used a slightly different but equivalent (assuming $\text{tr}(K_k) = 1$, $k = 1, \dots, K$) formulation without the $\beta$'s, which we introduced for illustration.

[2]It has been shown that solving semi-infinite problems like (7), using a method related to boosting (e.g. [8]) one requires at most $T = \mathcal{O}(\log(M)/\hat{\epsilon}^2)$ iterations, where $\hat{\epsilon}$ is the remaining constraint violation and the constants may depend on the kernels and the number of examples $N$ [11, 14]. At least for not too small values of $\hat{\epsilon}$ this technique produces reasonably fast good approximate solutions.

[3]Simplex based LP solvers often offer the possibility to efficient restart the computation when adding only a few constraints.

# References

[1] Francis R. Bach, Gert R. G. Lanckriet, and Michael I. Jordan. Multiple kernel learning, conic duality, and the SMO algorithm. In *Twenty-first international conference on Machine learning*. ACM Press, 2004.

[2] Kristin P. Bennett, Michinari Momma, and Mark J. Embrechts. Mark: a boosting algorithm for heterogeneous kernel models. *KDD*, pages 24–31, 2002.

[3] Jinbo Bi, Tong Zhang, and Kristin P. Bennett. Column-generation boosting methods for mixture of kernels. *KDD*, pages 521–526, 2004.

[4] O. Chapelle, V. Vapnik, O. Bousquet, and S. Mukherjee. Choosing multiple parameters for support vector machines. *Machine Learning*, 46(1-3):131–159, 2002.

[5] I. Grandvalet and S. Canu. Adaptive scaling for feature selection in SVMs. In *In Advances in Neural Information Processing Systems*, 2002.

[6] R. Hettich and K.O. Kortanek. Semi-infinite programming: Theory, methods and applications. *SIAM Review*, 3:380–429, September 1993.

[7] G.R.G. Lanckriet, T. De Bie, N. Cristianini, M.I. Jordan, and W.S. Noble. A statistical framework for genomic data fusion. *Bioinformatics*, 2004.

[8] R. Meir and G. Rätsch. An introduction to boosting and leveraging. In S. Mendelson and A. Smola, editors, *Proc. of the first Machine Learning Summer School in Canberra*, LNCS, pages 119–184. Springer, 2003. in press.

[9] C.S. Ong, A.J. Smola, and R.C. Williamson. Hyperkernels. In *In Advances in Neural Information Processing Systems*, volume 15, pages 495–502, 2003.

[10] J. Platt. Fast training of support vector machines using sequential minimal optimization. In B. Schölkopf, C.J.C. Burges, and A.J. Smola, editors, *Advances in Kernel Methods — Support Vector Learning*, pages 185–208, Cambridge, MA, 1999. MIT Press.

[11] G. Rätsch. *Robust Boosting via Convex Optimization*. PhD thesis, University of Potsdam, Computer Science Dept., August-Bebel-Str. 89, 14482 Potsdam, Germany, 2001.

[12] G. Rätsch, A. Demiriz, and K. Bennett. Sparse regression ensembles in infinite and finite hypothesis spaces. *Machine Learning*, 48(1-3):193–221, 2002. Special Issue on New Methods for Model Selection and Model Combination. Also NeuroCOLT2 Technical Report NC-TR-2000-085.

[13] G. Rätsch, S. Sonnenburg, and C. Schäfer. Learning interpretable svms for biological sequence classification. *BMC Bioinformatics, Special Issue from NIPS workshop on New Problems and Methods in Computational Biology Whistler, Canada, 18 December 2004*, 7(Suppl. 1:S9), February 2006.

[14] G. Rätsch and M.K. Warmuth. Marginal boosting. NeuroCOLT2 Technical Report 97, Royal Holloway College, London, July 2001.

[15] B. Schölkopf and A. J. Smola. *Learning with Kernels*. MIT Press, Cambridge, MA, 2002.

[16] S. Sonnenburg, G. Rätsch, and C. Schäfer. Learning interpretable SVMs for biological sequence classification. In *RECOMB 2005, LNBI 3500*, pages 389–407. Springer-Verlag Berlin Heidelberg, 2005.

[17] S. Sonnenburg, G. Rätsch, S. Schäfer, and B. Schölkopf. Large scale multiple kernel learning. *Journal of Machine Learning Research*, 2006. accepted.
